# Hyperparameters, Evidence and Generalisation for an Unrealisable Rule

**Glenn Marion and David Saad**
`glenny@ed.ac.uk, D.Saad@ed.ac.uk`
Department of Physics, University of Edinburgh,
Edinburgh, EH9 3JZ, U.K.

## Abstract

Using a statistical mechanical formalism we calculate the evidence, generalisation error and consistency measure for a linear perceptron trained and tested on a set of examples generated by a non linear teacher. The teacher is said to be unrealisable because the student can never model it without error. Our model allows us to interpolate between the known case of a linear teacher, and an unrealisable, nonlinear teacher. A comparison of the hyperparameters which maximise the evidence with those that optimise the performance measures reveals that, in the non-linear case, the evidence procedure is a misleading guide to optimising performance. Finally, we explore the extent to which the evidence procedure is unreliable and find that, despite being sub-optimal, in some circumstances it might be a useful method for fixing the hyperparameters.

## 1 INTRODUCTION

The analysis of supervised learning or learning from examples is a major field of research within neural networks. In general, we have a probabilistic[1] *teacher*, which maps an $N$ dimensional input vector $\mathbf{x}$ to output $y_t(\mathbf{x})$ according to some distribution $P(y_t \mid \mathbf{x})$. We are supplied with a data set $\mathcal{D} = (\{y_t(\mathbf{x}^\mu), \mathbf{x}^\mu\} : \mu = 1..p)$ generated from $P(y_t \mid \mathbf{x})$ by independently sampling the input distribution, $P(\mathbf{x})$, $p$ times. One attempts to optimise a model mapping (*a student*), parameterised by

some vector $\mathbf{w}$, with respect to the underlying teacher. The training error $E_{\mathbf{w}}(\mathcal{D})$ is some measure of the difference between the student and the teacher outputs over the set $\mathcal{D}$. Simply minimising the training error leads to the problem of *over-fitting*. In order to make successful predictions out-with the set $\mathcal{D}$ it is essential to have some prior preference for particular rules. *Occams razor* is an expression of our preference for the simplest rules which account for the data. Clearly $E_{\mathbf{w}}(\mathcal{D})$ is an unsatisfactory performance measure since it is limited to the training examples. Very often we are interested in the students ability to model a random example drawn from $P(y_t \mid \mathbf{x})P(\mathbf{x})$, but not necessarily in the training set, one measure of this performance is the generalisation error. It is also desirable to predict, or estimate, the level of this error. The teacher is said to be an *unrealisable rule*, for the student in question, if the minimum generalisation error is non-zero.

One can consider the Supervised Learning Paradigm within the context of Bayesian Inference. In particular MacKay [MacKay 92(a)] advocates the *evidence procedure* as a 'principled' method which, in some situations, does seem to improve performance [Thodberg 93]. However, in others, as MacKay points out the evidence procedure can be misleading [MacKay 92(b)].

In this paper we do not seek to comment on the validity of of the evidence procedure as an approximation to Hierarchical Bayes (see for example [Wolpert and Strauss 94]). Rather, we ask which performance measures do we seek to optimise and under what conditions will the evidence procedure optimise them? Theoretical results have been obtained for a linear perceptron trained on data produced by a linear perceptron [Bruce and Saad 94]. They suggest that the evidence procedure is a useful guide to optimising the learning algorithm's performance.

In what follows we examine the evidence procedure for the case of a linear perceptron learning a non linear teacher. In the next section we review the Bayesian scheme and define the evidence and the relevant performance measures. In section 3 we introduce our student and teacher and discuss the calculation. Finally, in section 4 we examine the extent to which the evidence procedure optimises performance.

## 2   BAYESIAN FORMALISM

### 2.1   THE EVIDENCE

If we take $E_{\mathbf{w}}(\mathcal{D})$ to be the usual sum squared error and assume that our data is corrupted by Gaussian noise with variance $1/2\beta$ then the probability, or *likelihood*, of the data($\mathcal{D}$) being produced given the model $\mathbf{w}$ and $\beta$ is $P(D \mid \beta, \mathbf{w}) \propto e^{-\beta E_{\mathbf{w}}(\mathcal{D})}$. In order to incorporate Occams Razor we also assume a prior distribution on the teacher rules, that is, we believe *a priori* in some rules more strongly than others. Specifically we believe that $P(\mathbf{w} \mid \gamma) \propto e^{-\gamma C(\mathbf{w})}$. Multiplying the likelihood by the prior we obtain the post training or student distribution[2] $P(\mathbf{w} \mid \mathcal{D}, \gamma, \beta) \propto e^{-\beta E_{\mathbf{w}}(\mathcal{D}) - \gamma C(\mathbf{w})}$. It is clear that the most probable model $\mathbf{w}^*$ is given by minimising the composite cost function $\beta E_{\mathbf{w}}(\mathcal{D}) + \gamma C(\mathbf{w})$ with respect to the weights ($\mathbf{w}$). This formalises the trade off between fitting the data and minimising student complexity. In this sense the Bayesian viewpoint coincides with the usual *backprop* standpoint.

In fact, it should be noted that stochastic minimisation can also give rise to the same post training distribution [Seung *et al* 92]. The parameters $\beta$ and $\gamma$ are known as the *hyperparameters*. Here we consider $C(\mathbf{w}) = \mathbf{w}^t\mathbf{w}$ in which case $\gamma$ is termed the weight decay.

The evidence is the normalisation constant in the above expression for the post training distribution.

$$P(\mathcal{D} \mid \gamma, \beta) = \int \prod_j dw_j \, P(\mathcal{D} \mid \beta, \mathbf{w}) P(\mathbf{w} \mid \gamma)$$

That is, the probability of the data set ($\mathcal{D}$) given the hyperparameters. The **evidence procedure** fixes the hyperparameters to the values that maximise this probability.

## 2.2 THE PERFORMANCE MEASURES

Many performance measures have been introduced in the literature (See *e.g.*, [Krogh and Hertz 92] and [Seung *et al* 92]). Here, we consider the squared difference between the average (over the post training distribution) of the student output $\langle y_s(\mathbf{x}) \rangle_{\mathbf{w}}$ and that of the teacher, $y_t(\mathbf{x})$, averaged over all possible test questions and teacher outputs, $P(y_t, \mathbf{x})$ and finally over all possible sets of data, $\mathcal{D}$.

$$\epsilon_g = \langle (y_t(\mathbf{x}) - \langle y_s(\mathbf{x}) \rangle_{\mathbf{w}})^2 \rangle_{P(\mathbf{x}, y_t), \mathcal{D}}$$

This is equivalent to the generalisation error given by Krogh and Hertz.

Another factor we can consider is the variance of the output over the student distribution $\langle \{y_s(\mathbf{x}) - \langle y_s(\mathbf{x}) \rangle_{\mathbf{w}}\}^2 \rangle_{\mathbf{w}, P(\mathbf{x})}$. This gives us a measure of the confidence we should have in our post training distribution and could possibly be calculated if we could estimate the input distribution $P(\mathbf{x})$. Here we extend Bruce and Saad's definition [Bruce and Saad 94] of the consistency measure $\delta_c$ to include unrealisable rules by adding the asymptotic error $\epsilon_g^{\infty} = \lim_{p \to \infty} \epsilon_g$,

$$\delta_c = \langle \{y_s(\mathbf{x}) - \langle y_s(\mathbf{x}) \rangle_{\mathbf{w}}\}^2 \rangle_{\mathbf{w}, P(\mathbf{x}), \mathcal{D}} - \epsilon_g + \epsilon_g^{\infty}$$

We regard $\delta_c = 0$ as optimal since then the variance over our student distribution is an accurate prediction of the decaying part of the generalisation error.

We can consider both these performance measures as *objective* functions measuring the students ability to mimic the underlying teacher. Clearly, they can only be calculated in theory and perhaps, estimated in practice. In contrast, the evidence is only a function of our assumptions and the data and the evidence procedure is, therefore, a practical method of setting the hyperparameters.

## 3 THE MODEL

In our model the student is simply a linear perceptron. The output for an input vector $\mathbf{x}^{\mu}$ is given by $y_s^{\mu} = \mathbf{w}.\mathbf{x}^{\mu}/\sqrt{N}$. The examples, against which the student is trained and tested, are produced by sampling the input distribution, $P(\mathbf{x})$ and then generating outputs from the distribution,

$$P(y_t \mid \mathbf{x}) = \sum_{\Omega=1}^{n} \frac{P(y_t \mid \mathbf{x}, \Omega) P(\mathbf{x} \mid \Omega) P_{\Omega}^t}{\sum_{\Omega=1}^{n} P(\Omega) P(\mathbf{x} \mid \Omega)}$$

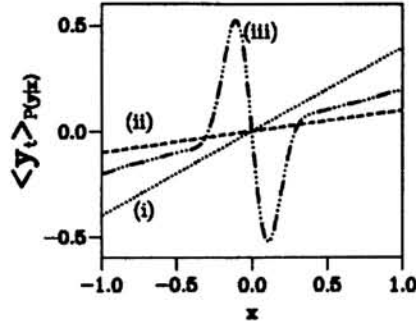

Figure 1: A 2-teacher in 1D : The average output $\langle y_t \rangle_{P(y|x)}$ (i) for $D_w = 0$ , (ii) for $D_w > 0$ ($\sigma_{x_1} = \sigma_{x_2}$) and (iii) with $D_w > 0$ ($\sigma_{x_1} \neq \sigma_{x_2}$).

where $P(y_t \mid \mathbf{x}, \Omega) \propto \exp([y_t - \mathbf{w}^\Omega.\mathbf{x}]^2/2\sigma^2)$, $P(\mathbf{x} \mid \Omega)$ is $N(\bar{a}_\Omega, \sigma_{x_\Omega}^2)$ [3] and $P_\Omega^t$ is chosen such that $\sum_{\Omega=1}^n P_\Omega^t = 1$. Thus, each component in the sum is a linear perceptron, whose output is corrupted by Gaussian noise of variance $\sigma^2$, and we refer to this teacher as an $n$-teacher.

In what follows, for simplicity, we consider a two teacher ($n$=2) with $\bar{a}_\Omega = 0$. The parameter $D_w = \frac{1}{N}|\mathbf{w}^1 - \mathbf{w}^2|^2$ and the input distribution determine the form of the teacher. This is shown in Figure 1. which displays the average output of a 2-teacher with one dimensional input vector. For $\sigma_{x_1} = \sigma_{x_2}$, $D_w$ controls the variance about a linear mean output, and for fixed $\sigma_{x_1} \neq \sigma_{x_2}$, $D_w$ controls the nonlinearity of the teacher. In the latter case, in the large $N$ limit the variance of $P(y_t \mid \mathbf{x})$ is zero.

We can now explicitly write the evidence and perform the integration over the student parameters (*over weights*). Taking the logarithm of the resulting expression leads to $\ln P(\mathcal{D} \mid \lambda, \beta) = -Nf(\mathcal{D})$ where the $f$ is analogous to a free energy in statistical physics.

$$-f(\mathcal{D}) = \frac{1}{2}\ln\frac{\lambda}{\pi} + \frac{\alpha}{2}\ln\frac{\beta}{\pi} + \frac{1}{2N}\ln det g + \frac{1}{2}\ln 2\pi + \frac{1}{N}\rho_j g_{jk}\rho_k - \Theta$$

and,

$$\rho_j = \{A_{jk}^\Omega w_k^\Omega + \frac{1}{\sqrt{N}}\eta^{\mu\Omega}x_j^{\mu\Omega}\}$$

$$\Theta = \frac{\beta}{N}\{A_{jk}^\Omega w_j^\Omega w_k^\Omega + \frac{2}{\sqrt{N}}\eta^{\mu\Omega}x_j^{\mu\Omega}w_j^\Omega + \eta^{\mu\Omega}\eta^{\mu\Omega}\}$$

$$g_{jk}^{-1} = \sum_{\Omega=1}^n A_{jk}^\Omega + \lambda\delta_{jk} \quad A_{jk}^\Omega = \frac{1}{N}x_j^{\mu\Omega}x_k^{\mu\Omega} \quad \lambda = \frac{\gamma}{\beta} \quad \alpha = \frac{p}{N}$$

Here we are using the convention that summations are implied where repeated indices occur.

The performance measures for this model are

$$\epsilon_g = \langle \frac{\sigma_{\Omega x}^2}{N} P_\Omega^t \{w_j^\Omega w_j^\Omega - 2w_j^\Omega \langle w_j \rangle_w + \langle w_j \rangle_w^2\} \rangle_\mathcal{D}$$

$$\delta_c = \frac{\sigma_{x\text{eff}}^2}{N\beta} \langle trg \rangle_\mathcal{D} - \epsilon_g + \epsilon_g^\infty$$

where,     $\langle w_j \rangle_w = \rho_k g_{kj}$·    and    $\sigma_{x\text{eff}}^2 = P_\Omega^t \sigma_{x_\Omega}^2$

In order to pursue the calculation we consider the average of $f(\mathcal{D})$ over all possible data sets just as, earlier, we defined our performance measures as averages over all data sets. This is some what artificial as we would normally be able to calculate $f(\mathcal{D})$ and be interested in the generalisation error for our learning algorithm given a particular instance of the data. However, here we consider the thermodynamic limit (i.e., $N, p \to \infty$ *s.t.* $\alpha = p/N = const.$) in which, due to our sampling assumptions, the behaviours for typical examples of $\mathcal{D}$ coincide with that of the average. Details of the calculation will be published else where [Marion and Saad 95].

## 4  RESULTS AND DISCUSSION

We can now examine the evidence and the performance measures for our unlearnable problem. We note that in two limits we recover the learnable, linear teacher, case. Specifically if the probability of picking one of the component teachers is zero or if both component teacher vectors are aligned. In what follows we set $P_1^t = P_2^t$ and normalise the components of the teacher such that $|\mathbf{w}^\Omega| = 1$.

Firstly let us consider the performance measures. The asymptotic value of both $\epsilon_g$ and $|\delta_c|$ for large $\alpha$ is $P_1^t P_2^t \sigma_{x_1}^2 \sigma_{x_2}^2 D_w / \sigma_{x\text{eff}}^2$. This is the minimum generalisation error attainable and reflects the effective noise level due to the mismatch between student and teacher.

We note here that the generalisation error is a function of $\lambda$ rather than $\beta$ and $\gamma$ independently. Figure 2a shows the generalisation error plotted against $\alpha$. The addition of unlearnability ($D_w > 0$) has a similar effect to the addition of noise on the examples. The appearance of the *hump* can be easily understood; If there is no noise or $\lambda$ is large enough then there is a steady reduction in $\epsilon_g$. However, if this is not so then for small $\alpha$ the student learns this effective noise and the generalisation error increases with $\alpha$. As the student gets more examples the effects of the noise begin to average out and the student starts to learn the rule. The point at which the generalisation error starts to decrease is influenced by the effective noise level and the prior constraint. Figure 2b shows the absolute value of the consistency measure v's $\alpha$ for non-optimal $\beta$. Again we see that unlearnability acts as an effective noise. For a few examples with $\lambda$ small or with large effective noise the student distribution is narrowed until the $\delta_c$ is zero. However, the generalisation error is still increasing (as described above) and $|\delta_c|$ increases to a local maximum, it then asymptotically tends to $\epsilon_g$. If there is no noise or $\lambda$ is large enough then $|\delta_c|$ steadily reduces as the number of examples increases.

We now examine the evidence procedure. Firstly we define $\beta_{ev}(\gamma)$ and $\gamma_{ev}(\beta)$ to be the hyperparameters which maximise the evidence. The evidence procedure

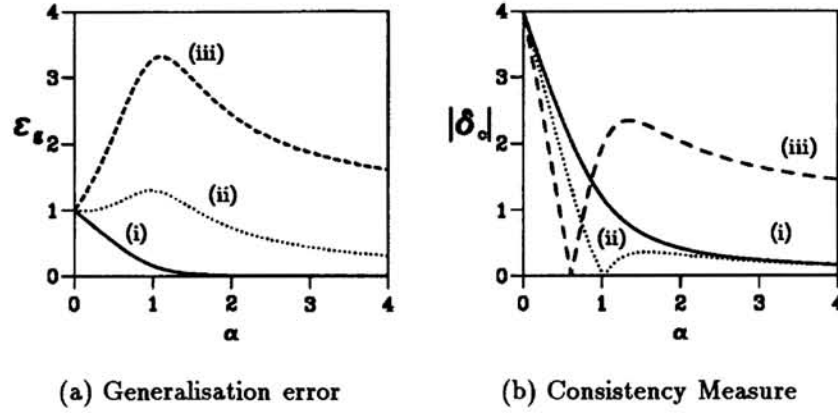

(a) Generalisation error          (b) Consistency Measure

Figure 2: The performance measures: Graph a shows $\epsilon_g$ for finite lambda. a(i) and a(ii) are the learnable case with noise in the latter case. a(iii) shows that the effect of *adding* unlearnability is qualitatively the same as adding noise. Graph b. shows the modulus of the consistency error v's $\alpha$. Curves b(i) and b(ii) are the learnable case without and with noise respectively. Curve b(iii) is an unlearnable case with the same noise level.

picks the point in hyperparameter space where these curves coincide. We denote the asymptotic values of $\beta_{ev}(\gamma)$ and $\gamma_{ev}(\beta)$ in the limit of large $\alpha$ by $\beta_\infty$ and $\gamma_\infty$ respectively.

In the linear case ($D_w = 0$) the evidence procedure assignments of the hyperparameters (for finite $\alpha$) coincide with $\beta_\infty$ and $\gamma_\infty$ and also optimise $\epsilon_g$ and $\delta_c$ in agreement with [Bruce and Saad 94] . This is shown in Figure 3a where we plot the $\beta$ which optimises the evidence ($\beta_{ev}$), the consistency measure ($\beta_{\delta_c}$) and the generalisation error ($\beta_{\epsilon_g}$) versus $\gamma$. The point at which the three curves coincide is the point in the $\beta$-$\gamma$ plane identified by the evidence procedure. However, we note here that, if one of the hyperparameters is poorly determined then maximising the evidence with respect to the other is a misleading guide to optimising performance even in the linear case.

The results for an unrealisable rule in the linear regime ($D_w > 0$, $\sigma_{x_1} = \sigma_{x_2}$) are similar to the learnable case but with an increased noise due to the unlearnability. The evidence procedure still optimises performance.

In the non-linear regime ($D_w > 0$ , $\sigma_{x_1} \neq \sigma_{x_2}$) the evidence procedure fails to minimise either performance measure. This is shown in Figure 3b where the evidence procedure point does not lie on $\beta_{\epsilon_g}(\gamma)$ or $\beta_{\delta_c}(\gamma)$. Indeed, its hyperparameter assignments do not coincide with $\beta_\infty$ and $\gamma_\infty$ but are $\alpha$ dependent.

How badly does the evidence procedure fail? We define the percentage degradation in generalisation performance as $K = 100 * (\epsilon_g(\lambda_{ev}) - \epsilon_g^{opt})/\epsilon_g^{opt}$. Where $\lambda_{ev}$ is the evidence procedure assignment and $\epsilon_g^{opt}$ is the optimal generalisation error with respect to $\lambda$. This is plotted in Figure 4a. We also define
$K_\delta = 100 * |\delta_c(\lambda_{ev})| / \epsilon_g(\lambda_{ev})$. This measures the error in using the variance of the

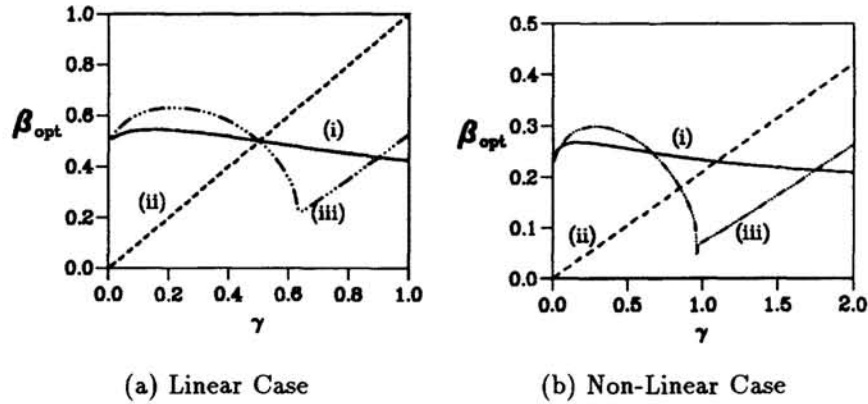

(a) Linear Case                                    (b) Non-Linear Case

Figure 3: The evidence procedure:Optimal $\beta$ v's $\gamma$. In both graphs for (i) the evidence($\beta_{ev}$), (ii) the generalisation error ($\beta_{\epsilon_g}$) and (iii) the consistency measure ($\beta_{\delta_c}$). The point which the evidence procedure picks in the linear case is that where all three curves coincide, whereas in the non linear case it coincides only with $\beta_{ev}$.

post training distribution to estimate the generalisation error as a percentage of the generalisation error itself. Examples of this quantity are plotted in Figure 4b. There are three important points to note concerning $\mathcal{K}$ and $\mathcal{K}_\delta$ . Firstly, the larger the deviation from a linear rule the greater is the error. Secondly, that it is the magnitude of the effective noise due to unlearnability relative to the real noise which determines this error. In other words, if the real noise is large enough to swamp the non-linearity of the rule then the evidence procedure will not be very misleading. Finally, the magnitude of the error for relatively large deviations from linearity is only a few percent and thus the evidence procedure might well be a reasonable, if not optimal, method for setting the hyperparameters. However, clearly it would be preferable to improve our student space to enable it to model the teacher.

## 5   CONCLUSION

We have examined the generalisation error, the consistency measure and the evidence procedure within a model which allows us to interpolate between a learnable and an unlearnable scenario. We have seen that the unlearnability acts like an effective noise on the examples. Furthermore, we have seen that for a linear student the evidence procedure breaks down, in that it fails to optimise performance, when the teacher output is non-linear. However, even for relatively large deviations of the teacher from linearity the evidence procedure is close to optimal.

Bayesian methods, such as the evidence procedure, are based on the assumption that the student or hypothesis space contains the teacher generating the data. In our case, in the non-linear regime, this is clearly not true and so it is perhaps not surprising that the evidence procedure is sub-optimal. Whether or not such a breakdown of the evidence procedure is a generic feature of a mismatch between the hypothesis space and the teacher is a matter for further study.

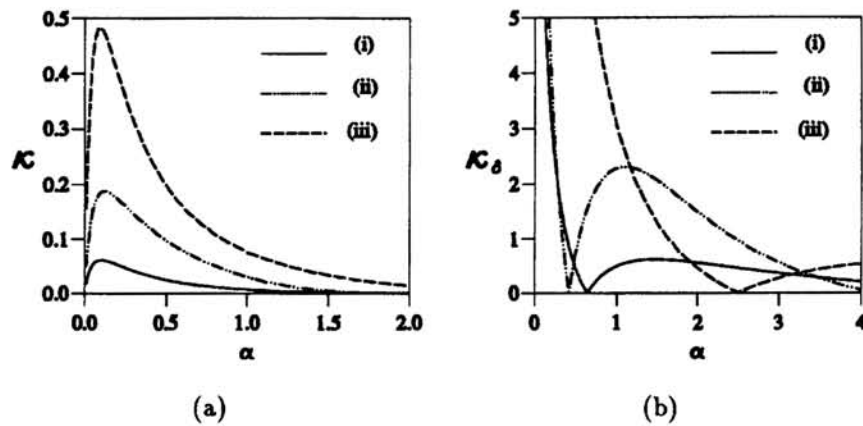

(a)                                          (b)

Figure 4: The relative degradation in performance compared to the optimal when using the evidence procedure to set the hyperparameters. Graph (a) shows the percentage degradation in generalisation performance $\kappa$. a(i) has $D_w = 1$ with the real noise level $\sigma = 1$. a(ii) has this noise level reduced to $\sigma = 0.1$ and a(iii) has increased non-linearity, $D_w = 3$, and $\sigma = 1$. Graph (b) shows the error made in predicting the generalisation error from the variance of the post training distribution as a percentage of the generalisation error itself, $\kappa_\delta$. b(i) and b(ii) have the same parameter values as a(i) and a(ii), whilst b(iii) has $D_w = 3$ and $\sigma = 0.1$

## Acknowledgments

We are very grateful to Alastair Bruce and Peter Sollich for useful discussions. GM is supported by an E.P.S.R.C. studentship.

## Footnotes

[1]This accommodates teachers with deterministic output corrupted by noise.

[2]Integrating this over $\beta$ and $\gamma$ gives us the posterior $P(\mathbf{w} \mid \mathcal{D})$.

[3]Where $N(\bar{x}, \sigma^2)$ denotes a normal distribution with mean $\bar{x}$ and variance $\sigma^2$.

## References

**Bruce, A.D. and Saad, D.** (1994) Statistical mechanics of hypothesis evaluation. *J. of Phys. A: Math. Gen.* **27**:3355-3363

**Krogh, A. and Hertz, J.** (1992) Generalisation in a linear perceptron in the presence of noise. *J. of Phys. A: Math. Gen.* **25**:1135-1147

**MacKay, D.J.C.** (1992a) Bayesian interpolation. *Neural Comp.* **4**:415-447

**MacKay, D.J.C.** (1992b) A practical Bayesian framework for backprop networks. *Neural Comp.* **4**:448-472

**Marion, G. and Saad, D.** (1995) A statistical mechanical analysis of a Bayesian inference scheme for an unrealisable rule. To appear in *J. of Phys. A: Math. Gen.*

**Seung, H. S, Sompolinsky, H., Tishby, N.** (1992) Statistical mechanics of learning from examples. *Phys. Rev. A*, **45**:6056-6091

**Thodberg, H.H.** (1994) Bayesian backprop in action:pruning, ensembles, error bars and application to spectroscopy. *Advances in Neural Information Processing Systems* **6**:208-215. Cowan *et al.*(Eds.), Morgan Kauffmann, San Mateo, CA

**Wolpert, D. H and Strauss, C. E. M.** (1994) What Bayes has to say about the evidence procedure. To appear in *Maximum entropy and Bayesian methods*. G. Heidbreder (Ed.), Kluwer.